# A Digital Antennal Lobe for Pattern Equalization: Analysis and Design

**Alex Holub, Gilles Laurent and Pietro Perona**
Computation and Neural Systems, California Institute of Technology
*holub@caltech.edu, laurentg@caltech.edu, perona@caltech.edu*

## Abstract

Re-mapping patterns in order to equalize their distribution may greatly simplify both the structure and the training of classifiers. Here, the properties of one such map obtained by running a few steps of discrete-time dynamical system are explored. The system is called 'Digital Antennal Lobe' (DAL) because it is inspired by recent studies of the antennal lobe, a structure in the olfactory system of the grasshopper. The pattern-spreading properties of the DAL as well as its average behavior as a function of its (few) design parameters are analyzed by extending previous results of Van Vreeswijk and Sompolinsky. Furthermore, a technique for adapting the parameters of the initial design in order to obtain opportune noise-rejection behavior is suggested. Our results are demonstrated with a number of simulations.

## 1 Introduction

The complexity of classifiers and the difficulty of learning their parameters is affected by the distribution of the input patterns. It is easier to obtain simple and accurate classifiers when the patterns associated with different classes are spaced far apart and evenly in the input space. Distributions which are lumpy, with classes bunched up in some regions of space leaving other regions of space empty may be more difficult to classify. This problem is particularly evident in sensory processing. In olfaction numerous odors which we wish to discriminate are chemically very similar, for example the citrus family (orange, lemon, lime...), while many odors that are in principle possible never occur in practice. The uneven chemical spacing for the odors of interest is expensive: in biological systems there is a premium in the simplicity of the classifiers that will recognize each individual odor.

When the dimension of the pattern space is large (e.g. $D > 100$), and the number of classes to be discriminated is relatively small (e.g. $N < 1000$), one may transform an uneven distribution of patterns into an evenly distributed one by means of a map that 'randomizes' the position of each pattern, i.e. that takes (small) neighborhoods of the input space and remaps them to random locations. In large-dimensional spaces it is exceedingly likely that two contiguous regions will be remapped to locations whose distance is comparable with the diameter of the space, and thus the distribution of patterns is equalized.

We explore a simple dynamical system which realizes one such map for spreading patterns in a high-dimensional space. The input space is the analog $D-$dimensional hypercube $(0,1)^D$ and the output space the digital hypercube $\{0,1\}^D$. The map is implemented by iterating a discrete-time first-order dynamical system consisting of two steps at each iteration: a first-order linear dynamical system followed by memoryless thresholding. The interest of the map is that it makes very parsimonious use of computational hardware (e.g. on the order of $D$ neurons or transistors) and yet it achieves good equalization in a few time steps. The ideas that we present are inspired by a computation that may take place in the olfactory system as suggested in Friedrichs and Laurent [1] and Laurent [2, 3]. In insects, the anatomical structure where this computation is presumed to take place is called the 'Antennal Lobe'. Because of this we call the map a 'Digital Antennal Lobe' (DAL).

## 2  The digital antennal lobe

The dynamical system we propose is inspired by the overall architecture of the antennal lobe and is designed to explore its computational capabilities. We apply two key simplifications: we discretize time into equally spaced 'epochs', updating synchronously the state of all the neurons in the network at each epoch, and we discretize the value of the state of each unit to the binary set $\{0,1\}$. The physiological justification for these simplifications goes beyond the scope of this paper.

Consider a collection of $N$ binary neurons which are randomly connected and updated synchronously. The network is initially quiescent (i.e. all the neurons have constant state zero). At some time an input is applied causing the network to take values that are different from zero. The state of the network evolves in time. The state of the network after a given constant number of time-steps (e.g. 10-20 time-steps) is the desired output of the system. Let us introduce the following notation:

| | |
|---|---|
| $N_E, N_I, N_U$ | Number of excitatory, inhibitory, and external input units. |
| $N$ | Total number of excitatory and inhibitory units ($N = N_E + N_I$) |
| $i$ | Neuron index: $i \in \{1, \ldots, N_E\}$ for excitatory and $i \in \{N_E + 1, \ldots, N\}$ for inhibitory. |
| $x_i^t \in \{0,1\} \forall i$ | Value of unit $i$ at time $t$. |
| $\vec{x}^t$ | Vector of values for all excitatory and inhibitory units at time $t$. |
| $c$ | Connectivity: $cN$ is the number of inputs to a given neuron. |
| $K_E, K_I, K_U$ | Excitatory, inhibitory, and external input (i.e. $K_E = cN_E$). |
| $A$ | Matrix of connections. $A$ has $cN^2$ nonzero entries. |
| $A_{ij}$ | Connection weight of unit $j$ to unit $i$. |
| $a_E, a_I, a_U$ | Excitatory, inhibitory, input weights ($A_{ij} \in \{a_I, 0, a_E\}$). |
| $\vec{\tau}$ | Activation thresholds for all the neurons |
| $\vec{u}$ | Vector of pattern inputs. |
| $B$ | Matrix of excitatory connections from pattern inputs to units. |
| $\vec{y}^t$ | Vector of neuronal input currents, i.e. $\vec{y}^{t+1} = A\vec{x}^t + B\vec{u}^t - \vec{\tau}$. |
| $\vec{x}^t = 1(\vec{y}^t)$ | Update equation for $\vec{x}$. $1(\cdot)$ is the Heaviside function. |
| $m^t$ | Mean activity in the network at time $t$, i.e. $m^t = \sum_i x_i / N$. |
| $m_u$ | Fraction of the external inputs which are active. |

A DAL may be generated once the value of 5 parameters are chosen. Assume excitatory connection weight $a_E = a_U = 1$ (this is a normalization constant). Choose a value for $a_I, c, \tau, N_I, N_E$. Generate random connection matrices $A$ and $B$ with average connectivity $c$ and connection weights $a_E, a_I$. Solve the following dynamical system forward in time from a zero initial condition:

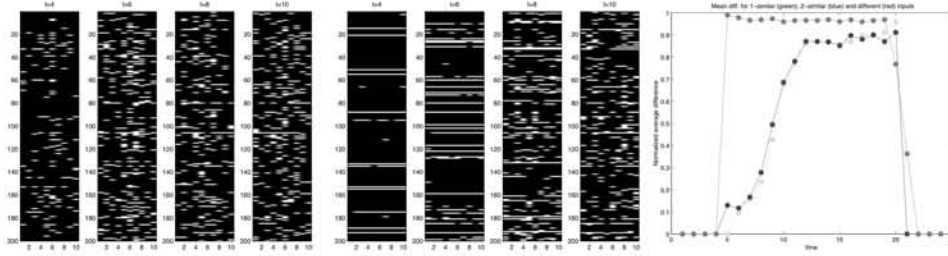

Figure 1: Example of pattern spreading by the a DAL. (Left) Response of a DAL to 10 uniformly distributed random olfactory input patterns applied at time epoch $t = 3$. Each vertical panel represents the state of excitatory units at a given time epoch (epochs $2, 4, 8, 10$ and excitatory units 1-200 are shown) in response to all stimuli. In a given panel the row index refers to a given excitatory unit and the column index to a given input pattern (200 of 1024 excitatory units shown and 10 input patterns). A white dot represents a state of '1' and a dark dot represent a state of '0'. Around 10% of the neurons are active (i.e. state = '1') by the $8^{th}$ time-epoch. The salt-and-pepper pattern present in each panel indicates that excitatory units respond differently to each input pattern. (Center) Activity of the DAL in response to 10 stimuli that differ only in one out of 1024 input dimensions, i.e. 0.1%. The horizontal streaks in the panels corresponding to early epochs ($t = 4$ and $t = 6$) indicate that the excitatory units respond equally or similarly to all input patterns. The salt-and-pepper pattern in later epochs indicates that the time course of each excitatory units state becomes increasingly different in time. (Right) Time-course of the normalized average distance between the patterns corresponding to different families of input patterns: the red curve corresponds to input patterns that are very different (average difference 20%), while the green and blue curve correspond to families of similar input patterns: 0.1% average difference for the green curve and 0.2% average difference for the blue curve. The parameters used in this network were $a_I = 10, c = .05, \tau = 10, N_E = 1024, N_I = 256$.

$$
\begin{aligned}
\vec{x}^0 &= \vec{0} & \text{zero initial condition} \\
\vec{y}^t &= A\vec{x}^{t-1} + B\vec{u} - \vec{\tau}, \quad t > 0 & \text{neuronal input} \\
\vec{x}^t &= 1(\vec{y}^t) & \text{state update}
\end{aligned}
$$

for some (constant) input pattern $\vec{u}$. The notation $1(\cdot)$ indicates the Heaviside step function.

The overall behavior of the DAL in response to different olfactory inputs is illustrated in Figure 1. Notice the main features of the DAL. (1) In response to an input each unit exhibits a complex temporal pattern of activity. (2) The pattern is different for different inputs. (3) The average activity rate of the neurons is approximately independent of the input pattern. (4) When very different input patterns are applied the average normalized Hamming distance between excitatory unit states is almost maximal immediately after the onset of the input stimulus. (5) When very similar input patterns are applied (e.g. 0.1% average difference), the average normalized Hamming distance between excitatory unit patterns is initially very small, i.e. initially the excitatory units respond similarly to similar inputs. The difference increases with time and reaches almost maximal value within 8-9 time-epochs.

The 'chaotic' properties of sparsely connected networks of neurons were noticed and studied by Van Vreeswijk and Sompolinsky [5] in the limit of $\infty$ neurons. In this paper we study networks with a small number of neurons comparable to the number observed within the antennal lobe. Additionally, we propose a technique

for the design of such networks, and demonstrate the possibility of 'stabilizing' some trajectories by parameter learning.

## 2.1 Analytic solution and equilibrium of network

The use of simplified neural elements, namely McCulloch-Pitts units [4], allows us to represent the system as a simple discrete time dynamical system. Furthermore, we are able to create expressions for various network properties. Several distributions can be used to approximate the number of active units in the population of excitatory, inhibitory, and external units, including: (1) the Binomial distribution, (2) the Poisson distribution, and (3) the Gaussian distribution. An approximation common to all three is that the activities of all units are uncorrelated. The Gaussian approximation will yield Van Vreeswijk and Sompolinsky's analysis [5].

Given the population activity at a time $t$, $m^t$, we can calculate the expected value for the population activity at the next time step, $m^{t+1}$:

$$E(m^{t+1}) = \sum_{e=0}^{K_E} \sum_{i=0}^{K_I} \sum_{u=0}^{K_U} p(e)p(i)p(u)1(a_E e + a_I i + a_U u - \tau)$$

Where $p(e)$, $p(i)$, and $p(u)$ are the probabilities of $e$ excitatory, $i$ inhibitory, and $u$ external inputs being active. Both $e$ and $i$ are binomially distributed with mean activity $m = m^t$, while the external input is binomially distributed with mean activity $m = m_u$:

$$p(j) = \binom{K_j}{j} (m)^j (1 - m)^{K_j - j}$$

The Poisson distribution can be used to approximate the binomial distribution for reasonable values of $\lambda$, where for instance $\lambda_e = K_e m^t$. Using the Poisson approximation, the probability of $j$ units being active is given by:

$$p(j) = \frac{e^{-\lambda_j}(\lambda_j)^j}{j!}$$

In the limit as $N \to \infty$, the distributions for the sum of the number of excitatory, inhibitory, and external units active approach normal distributions. Since the sum of Gaussian random variables is itself a Gaussian random variable, we can model the net input to a unit as the sum of the excitatory, inhibitory, and external input shifted by a constant representing the threshold. The mean $\mu$ and variance $\sigma^2$ of the Gaussian representing the input to an individual unit are then:

$$\mu = a_E m^t K_E + a_I m^t K_I + a_U m_u K_U - \tau$$
$$\sigma^2 = N_E[a_E^2 m^t c - a_E^2 c^2 m^t] + N_I[a_I^2 m^t c - a_I^2 c^2 m^t] + N_U[a_U^2 m_u c - a_U^2 c^2 m_u]$$

The fraction of active input units can be determined by considering the area under the gaussian corresponding to positive cumulative input:

$$m^{t+1} = 1 - \text{Erf}(-\frac{\mu}{\sigma})$$

The predicted population mean activity was calculated by imposing that the system is at equilibrium. The equilibrium condition is satisfied when $m^t = m^{t+1}$.

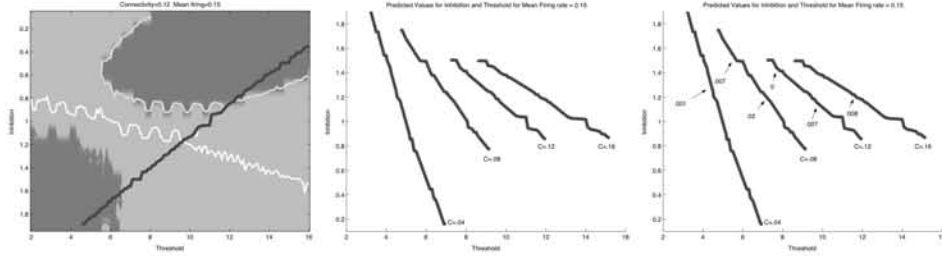

Figure 2: Design of a DAL. (Left) Behavior of the system for a given connectivity value. Light gray indicates inhibition-threshold values that yield a stable dynamical system. That is, small perturbations of firing activity do not result in large fluctuations in activity later in time. The dark blue line indicates equilibria, i.e. inhibition-threshold values for which the dynamical system rests at a constant mean-firing rate. (Center) The stable portions of the equilibrium curves for a number of connectivity values. Using this chart one may design an antennal lobe: for any given connectivity choose inhibition and threshold values that produce a desired mean firing rate. (Right) The design procedure produces networks that behave as desired. The arrows indicate parameter sets for which Monte Carlo simulation were performed in order to test the accuracy of the predictions. The values indexing the arrows correspond to the absolute difference of the predicted activity (.15) using a binomial approximation and the mean simulation activity across 10 random inputs to 10 different networks with the specified parameters sets.

We found the binomial approximation to yield the most accurate predictions in parameter ranges of interest to us, namely 500-4000 total units and connectivities ranging from .05-.15 (see Figure 2). The binomial approximation was always within 1 standard deviation of the Monte Carlo means. The Gaussian approximation yielded slightly less accurate predictions but required a fraction of the time to compute.

## 3    Design of the Antennal Lobe

The analysis described above allows us to design well behaved DALs. Specifically, we can predict which subsets of parameters in a given parameter range yield good network behavior. These predictions are made by solving the update equation for multiple sets of parameters and then determining which parameter ranges yield networks which are both stable and at equilibrium.

Figure 2 outlines the design technique for a network of 512 excitatory and 512 inhibitory units and a population mean activity of .15. The predicted activity of the network for different parameter sets corresponds well with that observed in Monte Carlo simulations. There is an average difference of .0061 between the predicted mean activity and that found in the simulations (see Figure 2, right plot).

## 4    Learning for trajectory stabilization

Consider a 'physical' implementation of the DAL, either by means of neurons in a biological system or by transistors in an electronic circuit. The inevitable presence of noise points to a fatal flaw of the DAL as we have seen it so far. The key property of the DAL is input decorrelation. In the presence of noise the same input applied multiple times to the same network will produce divergent trajectories, hence different final conditions, thus making the use of DALs for pattern

classification problematic.

Consider the possibility that noise is present in the system: as a result of fluctuations in the level of the input $\vec{u}$, fluctuations in the biophysical properties of the neurons, etc. We may represent this noise as an additional term $\vec{n}$ in the dynamical system:

$$
\begin{aligned}
\vec{y}^t &= A\vec{x}^t + B\vec{u}^t - \vec{\tau} \\
\vec{x}^{t+1} &= 1(\vec{y}^t + \vec{n}^t)
\end{aligned}
$$

Whatever the statistics of the noise, it is clear that it may influence the trajectory $\vec{x}$ of the dynamical system. Indeed, if $y_i^t$, the nominal input to a neuron, is sufficiently close to zero, then even a small amount of noise may change the state $x_i^t$ of that neuron. As we saw in earlier sections this implies that the ensuing trajectory will diverge from the trajectory of the same system with the same inputs and no noise or the same inputs and a different realization of the same noise process. This is shown in the left panel of Figure 3. On the other hand, if $y_i^t$ is far from zero, then $x_i^t$ will not change even with large amounts of noise. This raises the possibility that, if a DAL is appropriately designed, it may exhibit a high degree of robustness to noise. Ideally, for any given initial condition and input, and for any $\epsilon$, there exists a constant $y_0 > 0$ such that *any* initial condition and input in a $y_0$-ball around the original input and initial condition will produce trajectories that differ at most by $\epsilon$. Clearly, if $\epsilon = 0$ (i.e. the trajectory is required to be identical to the one of the noiseless system) then all trajectories of the system must coincide, not very useful. Similarly, if $\epsilon <\approx y_0$ the map will not spread different inputs. Therefore, this formulation of the problem does not have a satisfactory solution. One may, however, consider a weaker requirement. If the total number of patterns to be discriminated is not too large (probably 10-1000 in the case of olfaction) one could think of requiring noise robustness *only* for the trajectories $\vec{x}$ that are specific to those patterns. We therefore explored whether it was in principle possible to stabilize trajectories corresponding to different odor presentations rather than all trajectories.

We wish to change the connection weights $A, B$ and thresholds $T$ so that the network is robust with respect to noise around a given trajectory $\vec{x}(\vec{u})$. In order to achieve this we wish to ensure that at no time $t$ neuron $i$ has an input that is close to the threshold. If neuron $i$ is not firing at time $t$ (i.e. $x_i^t = 0$) then its input must be comfortably less than zero (i.e. for some constant $y_0 > 0$, $y_i^t < -y_0$) and viceversa for $x_i^t = 1$. We do so by minimizing an appropriate cost function: call $g(\cdot)$ an appropriate penalty function, e.g. $g(y) = \exp(y/y_0)$, then the cost of neuron $i$ at time $t$ if $x_i^t = 0$ is $C_i^t = g(y_i^t)$ and if $x_i^t = 1$ then $C_i^t = g(-y_i^t)$. Therefore:

$$
\begin{aligned}
C_i^t &= g((1 - 2x_i^t)y_i^t) \\
C(A, B, T) &= \sum_t \sum_i C_i^t
\end{aligned}
$$

The minimization may proceed by gradient descent. The equations for the gradient are:

$$
\begin{aligned}
\frac{\partial C_i^t}{\partial A_{ij}} &= \dot{g}((1 - 2x_i^t)y_i^t)(1 - 2x_i^t)\frac{\partial y_i^t}{\partial A_{ij}} \\
\frac{\partial y_i^t}{\partial A_{ij}} &= x_j^{t-1}
\end{aligned}
$$

similarly,

$$
\frac{\partial y_i^t}{\partial B_{ij}} = u_j^{t-1}
$$

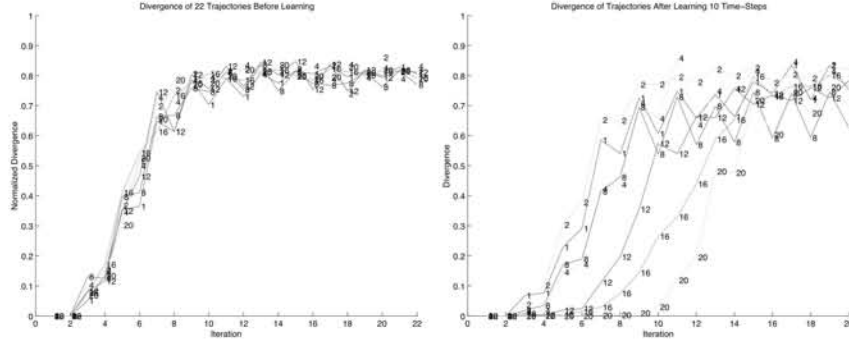

Figure 3: Robustness of trajectories to noise resulting from network learning. (Left) Pattern spreading in a DAL before learning. Each curve corresponds to the divergence rate between 10 identical trajectories in the presence of 5% gaussian synaptic noise added to each active presynaptic synapse. All patterns achieve maximum spreading in 9-10 steps as also shown in Figure 1. (Right) The divergence rate of the same trajectories after learning the first 10 steps of each trajectory. Each trajectory was learned sequentially, with the trajectory labelled 1 learned first. Note that trajectories learned later, for instance trajectory 20, diverge more slowly than earlier learned trajectories. Thus, the trajectories learned earlier are forgotten while more recently acquired trajectories are maintained. Furthermore, the trajectories maintain their stereotyped ability to decorrelate both after they are forgotten (e.g. trajectory 8) and after the 10 step learning period is over (e.g. trajectory 20). Untrained trajectories behave the same as trajectories in the left panel.

$$\frac{\partial y_i^t}{\partial T_i} = -1$$

In Figure 3 the results of one learning experiment are shown. Before learning all trajectories are susceptible to synaptic noise. After learning, those trajectories learned last exhibit robustness to noise, while trajectories learned earlier are slowly forgotten. We can compare each learned trajectory to a curve in multi-dimensional space with a 'robustness pipe' surrounding it. Any points lying within this pipe will be part of trajectories that remain within the pipe. In the case of olfactory processing, different odors correspond to unique trajectories, while trajectories lying within a common pipe correspond to the same input odor presentation.

A few details on the experiment: The network contained 2048 neurons, half of which were excitatory and the other half inhibitory. The values of the constants were: $c = 0.08, a_E = 1, a_I = 1.5, \tau = 7.2$, and the mean firing rate was set at about .05. The optimization took 60 gradient-descent steps.

## 5   Discussion and Conclusions

Sparsely connected networks of neurons have 'chaotic' properties which may be used for equalizing a set of patterns in order to make their classification easier. In studying the properties of such networks we extend previous results on networks with $\infty$ neurons by van Vreeswijk and Sompolinsky to the case of small number of neurons. We also provide techniques for designing networks that have desired average properties. Moreover, we propose a learning technique to make the network immune to noise around chosen trajectories while preserving the equalization property elsewhere.

A number of issues are left open. A precise characterization of the effects of the DAL on the distribution of the input parameters, and the consequent improvement in the ease of pattern classification is still missing. The geometry of the map implemented by the DAL is also unclear. Finally, it would be useful to obtain a quantitative estimate for the 'capacity' of the DAL, i.e. the number of trajectories which can be learned in any given network before older trajectories are forgotten.

## Acknowledgements

We would like to thank Or Neeman for useful suggestions and feedback. This work was supported in part by the Engineering Research Centers Program of the National Science Foundation under Award Number EEC-9402726.

## References

[1] Friedrich R. & Laurent, G. (2001) Dynamical optimization of odor representations by slow temporal patterning of mitral cell activity. *Science* **291**:889-894.

[2] Laurent G, Stopfer M, Friedrich RW, Rabinovich MI, Volkovskii A, Abarbanel HD. (2001) Odor encoding as an active, dynamical process: experiments, computation, and theory. *Ann Rev Neurosci.* **24**:263-97.

[3] Laurent G. (2002) Olfactory network dynamics and the encoding of multidimensional signals. *Nat Rev Neurosci* **3**(11):884-95.

[4] McCulloch WS, Pitts W. (1943). A logical calculus of ideas immanent in nervous activity. *Bulletin of Mathematical Biophysics* **5**: 115-133.

[5] van Vreeswijk C, Sompolinsky H (1998) Chaotic balanced state in a model of cortical circuits. *Neural Computation.* **10**(6):1321-71.
